# Speakers optimize information density through syntactic reduction

**Roger Levy**
Department of Linguistics
UC San Diego
9500 Gilman Drive
La Jolla, CA 92093-0108, USA
rlevy@ling.ucsd.edu

**T. Florian Jaeger**
Department of Linguistics & Department of Psychology
Stanford University & UC San Diego
9500 Gilman Drive
La Jolla, CA 92093-0109, USA
tiflo@csli.stanford.edu

## Abstract

If language users are rational, they might choose to structure their utterances so as to optimize communicative properties. In particular, information-theoretic and psycholinguistic considerations suggest that this may include maximizing the uniformity of information density in an utterance. We investigate this possibility in the context of *syntactic reduction*, where the speaker has the option of either marking a higher-order unit (a phrase) with an extra word, or leaving it unmarked. We demonstrate that speakers are more likely to reduce less information-dense phrases. In a second step, we combine a stochastic model of structured utterance production with a logistic-regression model of syntactic reduction to study which types of cues speakers employ when estimating the predictability of upcoming elements. We demonstrate that the trend toward predictability-sensitive syntactic reduction (Jaeger, 2006) is robust in the face of a wide variety of control variables, and present evidence that speakers use both surface and structural cues for predictability estimation.

## 1  Introduction

One consequence of the expressive richness of natural languages is that usually more than one means exists of expressing the same (or approximately the same) message. As a result, speakers are often confronted with choices as to how to structure their intended message into an utterance. At the same time, linguistic communication takes place under a host of cognitive and environmental constraints: speakers and addressees have limited cognitive resources to bring to bear, speaker and addressee have incomplete knowledge of the world and of each other's state of knowledge, the environment of communication is noisy, and so forth. Under these circumstances, if speakers are rational then we can expect them to attempt to optimize the communicative properties of their utterances.

But what are the communicative properties that speakers choose to optimize? The prevalence of ambiguity in natural language—the fact that many structural analyses are typically available for a given utterance—might lead one to expect that speakers seek to minimize structural ambiguity, but both experimental (Arnold et al., 2004, inter alia) and corpus-based (Roland et al., 2006, inter alia) investigations have found little evidence for active use of ambiguity-avoidance strategies. In this paper we argue for a different locus of optimization: that speakers structure utterances so as to optimize *information density*. Here we use the term "information" in its most basic information-theoretic sense—the negative log-probability of an event—and by "information density" we mean the amount of information per unit comprising the utterance. If speakers behave optimally, they should structure their utterances so as to avoid peaks and troughs in information density (see also (Aylett and Turk, 2004; Genzel and Charniak, 2002)). For example, this principle of uniform information density (UID) as an aspect of rational language production predicts that speakers should modulate phonetic

duration in accordance with the predictability of the unit expressed. This has been shown by Bell et al. (2003, inter alia) for words and by Aylett and Turk (2004) for syllables. If UID is a general principle of communicative optimality, however, its effects should be apparent at higher levels of linguistic production as well. In line with this prediction are the results of Genzel and Charniak (2002); Keller (2004), who found that sentences taken out of context have more information the later they occur in a discourse. For phonetic reduction, choices about word duration can directly modulate information density. However, it is less clear how the effects of UID at higher levels of language production observed by Genzel and Charniak (2002) and Keller (2004) come about. Genzel and Charniak (2002) show that at least part of their result is driven by the repetition of open-class words, but it is unclear how this effect relates to a broader range of choice points within language production. In particular, it is unclear whether any choices above the lexical level are affected by information density (as expected if UID is general). In this paper we present the first evidence that speakers' choice during syntactic planning is affected by information density optimization. This evidence comes from *syntactic reduction*—a phenomenon in which speakers have the choice of either marking a phrase with an optional word, or leaving it unmarked (Section 3). We show that in cases where the phrase is marked, the marking reduces the phrase's information density, and that the phrases that get marked are the ones that would otherwise be the most information-dense (Section 4). This provides crucial support for UID as a general principle of language production.

The possibility that speakers' use of syntactic reduction optimizes information density leads to questions as to how speakers estimate the probability of an upcoming syntactic event. In particular, one can ask what types of cues language users employ when estimating these probabilites. For example, speakers could compute information density using only surface cues (such as the words immediately preceding a phrase). On the other hand, they might also take structural features of the utterance into account. We investigate these issues in Section 5 using an incremental model of structured utterance production. In this model, the *predictability* of the upcoming phrase markable by the optional word is taken as a measure of the phrase's information density. The resulting predictability estimate, in turn, becomes a covariate in a separate model of syntactic reduction. Through this two-step modeling approach we show that predictability is able to explain a significant part of the variability in syntactic reduction, and that evidence exists for speakers using both structural and surface cues in estimating phrasal predictability.

## 2   Optimal information density in linguistic utterances

We begin with the information-theoretic definition that the information conveyed by a complete utterance $u$ is $u$'s Shannon information content (also called its *surprisal*), or $\log_2 \frac{1}{P(u)}$. If the complete utterance $u$ is realized in $n$ units (for example, words $w_i$), then the information conveyed by $u$ is the sum of the information conveyed by each unit of $u$:

$$\log \frac{1}{P(u)} = \log \frac{1}{P(w_1)} + \log \frac{1}{P(w_2|w_1)} + \cdots + \log \frac{1}{P(w_n|w_1 \cdots w_{n-1})} \qquad (1)$$

For simplicity we assume that each $w_i$ occupies an equal amount of time (for spoken language) or space (written language). Optimization of information density entails that the information conveyed by each $w_i$ should be as uniform and close to an ideal value as possible. There are at least two ways in which UID may be optimal. First, the transmission of a message via spoken or written language can be viewed as a noisy channel. From this assumption it follows that information density is optimized near the channel capacity, where speakers maximize the rate of information transmission while minimizing the danger of a mistransmitted message (see also Aylett (2000); Aylett and Turk (2004); Genzel and Charniak (2002)). That is, UID is an optimal solution to the problem of *rapid yet error-free communication* in a noisy environment.

Second and independently of whether linguistic communication is viewed as a noisy channel, UID can be seen as minimizing comprehension difficulty. The difficulty incurred by a comprehender in processing a word $w_i$ is positively correlated with its surprisal (Hale, 2001; Levy, 2006). If the effect of surprisal on difficulty is superlinear, then the total difficulty of the utterance $u$ ($\sum_{i=1}^{n} [\log \frac{1}{P(w_i|w_1 \cdots w_{i-1})}]^k$ with $k > 1$) is minimized when information density is uniform (for

proof see appendix; see also Levy 2005, ch. 2).[1] That is, UID is also an optimal solution to the problem of *low-effort comprehension*.

# 3 Syntactic reduction

UID would be optimal in several ways, but do speakers actually consider UID as a factor when making choices during online syntactic production? We address this question by directly linking a syntactic choice point to UID. If information density optimization is general, i.e. if it applies to all aspects of language production, we should find its effects even in structural choices.

We use variation in the form of certain types of English relative clauses (henceforth RCs) to test this hypothesis. At the onset of an RC speakers can, but do not have to, utter the relativizer *that*.[2] We refer to the omission of *that* as syntactic REDUCTION.

(1)     How big is [$_{NP}$ the family$_i$ [$_{RC}$ (*that*) you cook for $_{-i}$ ]]?

Our dataset consists of a set of 3,452 RCs compatible with the above variation, extracted from the Switchboard corpus of spontaneous American English speech. All RCs were automatically annotated for a variety of control factors that are known to influence syntactic reduction of RCs, including RC size, distance of the RC from the noun it modifies, data about the speaker including gender and speech rate, local measures of speech disfluency, and formal and animacy properties of the RC subject (a full list is given in the appendix; see also (Jaeger, 2006)). These control factors are used in the logistic regression models presented in Section 5.

# 4 Reduction as a means of information density modulation

From a syntactic perspective, the choice to omit a relativizer means that the first word of an RC conveys two pieces of information simultaneously: the onset of a relative clause and part of its internal contents (usually part of its subject, as *you* in Example (1)). Using the notation $w_{...-1}$ for the context preceding the RC and $w_1$ for the RC's first word (excluding the relativizer, if any), these two pieces of information can be expressed as a Markov decomposition of $w_1$'s surprisal:

$$\log \frac{1}{P(w_1|w_{...-1})} = \log \frac{1}{P(\text{RC}|w_{...-1})} + \log \frac{1}{P(w_1|\text{RC}, w_{...-1})} \qquad (2)$$

Conversely, the choice to use a relativizer separates out these two pieces of information, so that the only information carried by $w_1$ is measured as

$$\log \frac{1}{P(w_1|\text{RC}, \textit{that}, w_{...-1})} \qquad (3)$$

If the overall distribution of syntactic reduction is in accordance with principles of information-density optimization, we should expect that full forms (overt relativizers) should be used more often when the information density of the RC *would be high if the relativizer were omitted.* The information density of the RC and subsequent parts of the sentence can be quantified by their Shannon information content. As a first test of this prediction, we use $n$-gram language models to measure the relationship between the Shannon information content of the first word of an RC and the tendency toward syntactic reduction.

We examined the relationship between rate of syntactic reduction and the surprisal that $w_1$ would have if no relativizer had been used—that is, $\log \frac{1}{P(w_1|w_{...-1})}$—as estimated by a trigram language

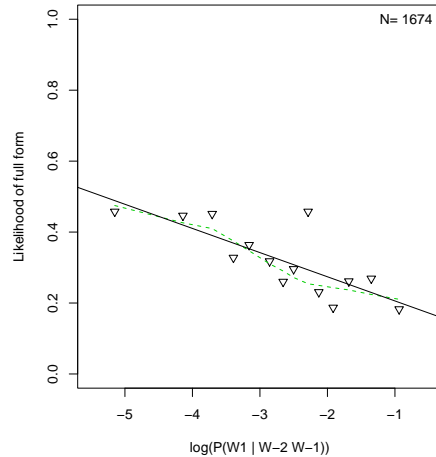

Figure 1: RC $n$-gram-estimated information density and syntactic reduction. Dotted green line indicates lowess fit.

model.[3] To eliminate circularity from this test (the problem that for an unreduced RC token, $P(w_1|w_{...-1})$ may be low precisely because *that* is normally inserted between $w_{...-1}$ and $w_1$), we estimated $P(w_1|w_{...-1})$ from a version of the Switchboard corpus in which all optional relativizers were omitted. That is, if we compare actual English with a hypothetical pseudo-English differing only in the absence of optional relativizers, are the overt relativizers in actual English distributed in a way such that they occur more in the contexts that would be of highest information density in the pseudo-English?[4] For every actual instance of an RC onset $\cdots w_{-2}w_{-1}(that)w_1 \cdots$ we calculated the trigram probability $P(w_1|w_{-2}w_{-1})$: that is, an estimate of the probability that $w_1$ would have *if no relativizer had been used*, regardless of whether a relativizer was actually used or not. We then examined the relationship between this probability and the outcome event: whether or not a relativizer was actually used. Figure 4 shows the relationship between the different quantiles of the log-probability of $w_1$ and the likelihood of syntactic reduction. As can be seen, reduction is more common when the probability $P(w_1|w_{-n} \cdots w_{-1})$ is high. This inverse relationship between $w_1$ surprisal and relativizer use matches the predictions of UID. [5]

## 5  Structural predictability and speaker choice

Section 4 provides evidence that speakers' choices about syntactic reduction are correlated with information density: RC onsets that would be more informationally dense in reduced form are less likely to be reduced. This observation does not, however, provide strong evidence that speakers are directly sensitive to information density in their choices about reduction. Furthermore, if speakers *are* sensitive to information density in their reduction choices, it raises a new question: what kind of information is taken into account in speakers' estimation of information density?

This section addresses the questions of whether reduction is directly sensitive to information density, and what information might be used in estimates of information density, using a two-step modeling approach. The first step involves a incremental stochastic model of structured utterance production. This model is used to construct estimates of the first term in Equation (2) contributing to an RC onset's information density: the *predictability* (conditional probability) of an RC beginning at a

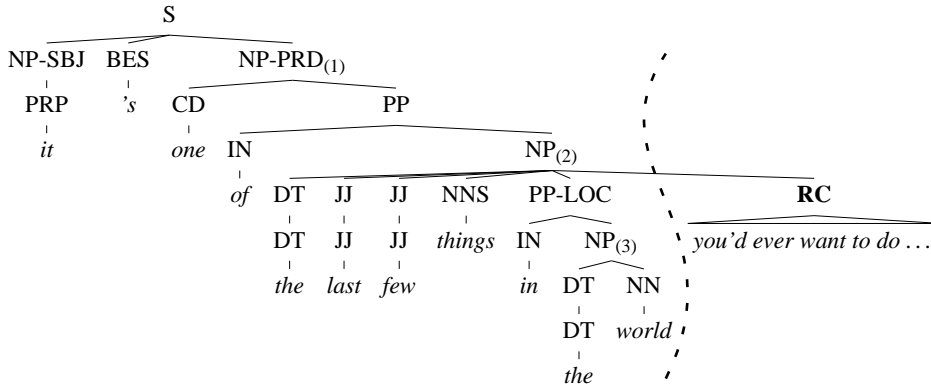

Figure 2: A flattened-tree representation of a sentence containing an RC. The incremental parse through *world* consists of everything to the left of the dashed line.

given point in a sentence, given an incremental structural representation for the sentence up to that point. Because the target event space of this term is small, a wide variety of cues, or features, can be included in the model, and the reliability of the resulting predictability estimates is relatively high. This model is described in Section 5.1. The resulting predictability estimates serve as a crucial covariate in the second step: a logistic regression model including a number of control factors (see Section 3 and appendix). This model is used in Sections 5.3 as a stringent test of the explanatory power of UID for speakers' reduction choices, and in Section 5.4 to determine whether evidence exists for speakers using structural as well as surface cues in their predictability estimates.

## 5.1 A structural predictability model

In this section we present a method of estimating the predictability of a relative clause in its sentential context, contingent on the structural analysis of that context. For simplicity, we assume that structural analyses are context-free trees, and that the complete, correct incremental analysis of the sentential context is available for conditioning.[6] In general, the task is to estimate

$$P(RC_{n+1...}|w_{1...n}, T_{1...n}) \tag{4}$$

that is, the probability that a phrase of type RC appears in the utterance beginning at $w_{n+1}$, given the incremental structured utterance $\langle w_{1...n}, T_{1...n} \rangle$. To estimate these probabilities, we model production as a fully incremental, top-down stochastic tree generation process similar to that used for parsing in Roark (2001). Tree production begins by expanding the root node, and the expansion process for each non-terminal node $N$ consists of the following steps:

(a) choosing a leftmost daughter event $D_1$ for $N$, and making it the *active* node;

(b) recursively expanding the active node; and

(c) choosing the next right-sister event $D_{i+1}$, and making it the active node.

Steps (b) and (c) are repeated until a special right-sister event $*END*$ is chosen in step (c), at which point expansion of $N$ is complete. As in Collins (2003) and Roark (2001), this type of directed generative process allows conditioning on arbitrary features of the incremental utterance.

After each word $w_n$, the bottom-right preterminal of the incremental parse is taken as the currently active node $N_0$; if its $i$-th ancestor is $N_i$ then we have:[7]

$$P(RC_{n+1...}|w_{1...n}, T_{1...n}) = \sum_{i=0}^{k} \left[ P(RC|N_i) \prod_{j=0}^{i-1} P(*END*|N_j) \right] \qquad (5)$$

Figure 2 gives an example of an incremental utterance just before an RC, and illustrates how Equation (5) might be applied.[8] At this point, NN would be the active node, and step (b) of expanding $NP_{(3)}$ would have just been completed. An RC beginning after $w_n$ (*world* in Figure 2) could conceivably modify any of the NPs marked (1)-(3), and all three of those attachments may contribute probability mass to $P(RC_{n+1...})$, but an attachment at $NP_{(2)}$ can only do so if $NP_{(1)}$ and PP-LOC make no further expansion.

## 5.2   Model parameters and estimation

What remains is to define the relevant event space and estimate the parameters of the tree-generation model. For RC predictability estimation, the only relevant category distinctions are between RC, $*END*$, and any other non-null category, so we limit our event space to these three outcomes. Furthermore, because RCs are never leftmost daughters, we can ignore the parameters determining first-daughter event outcome probabilities (step (a) in Section 5.1). We estimate event probabilities using log-linear models (Berger et al., 1996; Della Pietra et al., 1997).[9]

We included five classes of features in our models, chosen by linguistic considerations of what is likely to help predict the next event given an active node in an incremental utterance (see Wasow et al. (ress)):

- NGRAM features: the last one, two, and three words in the incremental utterance;

- HEAD features: the head word and head part of speech (if yet seen), and animacy (for NPs) of the currently expanded node;

- HISTory features: the incremental constituent structure of the currently expanded node $N$, and the number of words and sister nodes that have appeared to the right of $N$'s head daughter;

- PRENOMinal features: when the currently expanded node is an NP, the prenominal adjectives, determiners, and possessors it contains;

- EXTernal features: when the currently expanded node is an NP, its external grammatical function, and the verb in the clause it governs.

The complete set of features used is listed in a supplementary appendix.

[NP [NP something else] [RC we could have done]]

Tree canonicalization consisted of ensuring that each phrasal node had a preterminal head daughter, and that each preterminal node headed a phrasal node, according to the head-finding algorithm of Collins (2003). VP and S nodes without a verbal head child were given special null-copula head daughters, so that the NP-internal constituency of predicative nouns without overt copulas was distinguished from sentence-level constituency.

[9]The predictability models were heavily overparameterized, and to prevent overfitting were regularized with a quadratic Bayesian prior. For each trained model the value of the regularization parameter (constant for all features) was chosen to optimize held-out data likelihood. RC probabilities were estimated using ten-fold cross-validation over the entire Switchboard corpus, so that a given RC was never contained in the training data of the model used to determine its probability.

### 5.3 Explanatory power of phrasal predictability

We use the same statistical procedures as in (Jaeger, 2006, Chapter 4) to put the predictions of the information-density hypothesis to a more stringent test. We evaluate the explanatory power of phrasal predictability in logistic regression models of syntactic reduction that include all the control variables otherwise known to influence relativizer omission (Section 3). To avoid confounds due to clusters of data points from the same speaker, the model was bootstrapped $(10, 000$ iterations) with random replacement of speaker clusters.[10] Phrasal predictability of the RC (based on the full feature set listed in Section 5.2) was entered into this model as a covariate to test whether RC predictability co-determines syntactic reduction after other factors are controlled for. Phrasal predictability makes a significant contribution to the relativizer omission model $(\chi^2(1) = 54.3; p < 0.0001)$. This demonstrates that phrasal predictability has explanatory power in this case of syntactic reduction.

### 5.4 Surface and structural conditioning of phrasal predictability

The structural predictability model puts us in a position to ask whether empirically observed patterns of syntactic reduction give evidence for speakers' use of some types of cues but not others. In particular, there is a question of whether predictability based on surface cues alone (the NGRAM features of Section 5.2) provides a complete description of information-density effects on speakers' choices in syntactic reduction. We tested this by building a syntactic-reduction model containing two predictability covariates: one using NGRAM features alone, and one using all other (i.e., structural, or all-but-NGRAM) feature types listed in Section 5.2. We can then test whether the parameter weight in the reduction model for each predictability measure differs significantly from zero. It turns out that both predictability measures matter: all-but-NGRAM predictability is highly significant $(\chi^2(1) = 23.55, p < 0.0001)$, but NGRAM predictability is also significant $(\chi^2(1) = 5.28, p < 0.025)$. While NGRAM and all-but-NGRAM probabilities are strongly correlated $(r^2 = 0.70)$, they evidently exhibit enough differences to contribute non-redundant information in the reduction model. We interpret this as evidence that speakers may be using both surface and structural cues for phrasal predictability estimation in utterance structuring.

## 6 Conclusion

Using a case study in syntactic reduction, we have argued that information-density optimization—the tendency to maximize the uniformity of upcoming-event probabilities at each part of a sentence—plays an important role in speakers' choices about structuring their utterances. This question has been previously addressed in the context of phonetic reduction of highly predictable words and syllables (Aylett and Turk, 2004; Bell et al., 2003), but not in the case of word reduction. Using a stochastic tree-based model of incremental utterance production combined with a logistic regression model of syntactic reduction, we have found evidence that when speakers have the choice between using or omitting an optional function word that marks the onset of a phrase, they use the function word more often when the phrase it introduces is less predictable. We have found evidence that speakers may be using both phrasal and structural information to calculate upcoming-event predictabilities. The overall distribution of syntactic reduction has the effect of smoothing the information profile of the sentence: when the function word is not omitted, the information density of the immediately following words is reduced. The fact that our case study involves the omission of a single word with little to no impact on utterance meaning made the data particularly amenable to analysis, but we believe that this method is potentially applicable to a wider range of variable linguistic phenomena, such as word ordering and lexical choice.

More generally, we believe that the ensuing view of constraints on situated linguistic communication as limits on the information-transmission capacity of the environment, or on information-processing capacity of human language processing faculties, can serve as a useful framework for the study of

language use. On this view, syntactic reduction is available to the speaker as a pressure valve to regulate information density when it is dangerously high. Equivalently, the presence of a function word can be interpreted as a signal to the comprehender to expect the unexpected, a rational exchange of time for reduced information density, or a meaningful delay (Jaeger, 2005). More generally, reduction at different levels of linguistic form (phonetic detail, detail of referring expressions, as well as omission of words, as in the case examined here) provides a means for speakers to smooth the information-density profile of their utterances (Aylett and Turk, 2004; Genzel and Charniak, 2002). This raises important questions about the specific motivations of speakers' choices: are these choices made for the sake of facilitating production, or as part of audience design? Finally, this view emphasizes the connection between grammatical optionality and communicative optimality. The availability of more than one way to express a given meaning grants speakers the choice to select the optimal alternative for each communicative act.

### Acknowledgments

This work has benefited from audience feedback at the Language Evolution and Computation research group at the University of Edinburgh, and at the Center for Research on Language at UC San Diego. The idea to derive estimates of RC predictability based on multiple cues originated in discussion with T. Wasow, P. Fontes, and D. Orr. RL's work on this paper was supported by an ESRC postdoctoral fellowship at the School of Informatics at the University of Edinburgh (award PTA-026-27-0944). FJ's work was supported by a research assistantship at the Linguistics Department, Stanford University (sponsored by T. Wasow and D. Jurafsky) and a post-doctoral fellowship at the Department of Psychology, UC San Diego (V. Ferreira's NICHD grant R01 HD051030).

## Footnotes

[1] Superlinearity would be a natural consequence of limited cognitive resources, although the issue awaits further empirical investigation.

[2] To be precise, standard American English restricts omission of *that* to finite, restrictive, non-pied-piped, non-extraposed, non-subject-extracted relative clauses. Only such RCs are considered here.

[3]In cases where the conditioning bigram was not found, we backed off to a conditioning unigram, and omitted cases where the conditioning unigram could not be found; no other smoothing was applied. We used hold-one-out estimation of $n$-gram probabilities to prevent bias.

[4]Omitting optional relativizers in the language model can alternatively be interpreted as assuming that speakers equate (3) with the second term of (2)—that is, the presence or absence of the relativizer is ignored in estimating the probablity of the first word of a relative clause.

[5]We also calculated the relationship for estimates of RC information density using a trigram model of the Switchboard corpus as-is. By this method, there *is* a priori reason to expect a correlation, and indeed reduction is (more strongly than in Figure 4) negatively correlated with this measure.

[6]If predictability from the perspective of the comprehender rather than the producer is taken to be of primary interest, this assumption may seem controversial. Nevertheless, there is little evidence that incremental structural misanalysis is a pervasive phenomenon in naturally occuring language (Roland et al., 2006), and the types of incremental utterances occurring immediately before relative clauses do not seem to be good candidates for local misanalysis. From a practical perspective, assuming access to the correct incremental analysis avoids the considerable difficulty involved in the incremental parsing of speech.

[7]Equation (5) relies on the fact that an RC can never be the first daughter of a node expansion; the possibility of RC generation through left-recursion can thus be ignored.

[8]The phrase structures found in the Penn Treebank were flattened and canonicalized to ensure that the incremental parse structures do not contain implicit information about upcoming constituents. For example, RC structures are annotated with a nested NP structure, such as

[10]Our data comes from approximately 350 speakers contributing 1 to 40 RCs (MEAN= 10, MEDIAN= 8, SD= 8.5) to the data set. Ignoring such clusters in the modeling process would cause the models to be overly optimistic. Post-hoc tests conducted on the models presented here revealed no signs of over-fitting, which means that the model is likely to generalize beyond the corpus to the population of American English speakers. The significance levels reported in this paper are based on a normal-theory interpretation of the unbootstrapped model parameter estimate, using a bootstrapped estimate of the parameter's standard error.

# References

Arnold, J. E., Wasow, T., Asudeh, A., and Alrenga, P. (2004). Avoiding attachment ambiguities: The role of constituent ordering. *Journal of Memory and Language*, 51:55–70.

Aylett, M. (2000). *Stochastic Suprasegmentals: Relationships between Redundancy, Prosodic Structure and Care of Articulation in Spontaneous Speech*. PhD thesis, University of Edinburgh.

Aylett, M. and Turk, A. (2004). The Smooth Signal Redundancy Hypothesis: A functional explanation for relationships between redundancy, prosodic prominence, and duration in spontaneous speech. *Language and Speech*, 47(1):31–56.

Bell, A., Jurafsky, D., Fosler-Lussier, E., Girand, C., Gregory, M., and Gildea, D. (2003). Effects of disfluencies, predictability, and utterance position on word form variation in English conversation. *Journal of the Acoustical Society of America*, 113(2):1001–1024.

Berger, A. L., Pietra, S. A. D., and Pietra, V. J. D. (1996). A Maximum Entropy approach to natural language processing. *Computational Linguistics*, 22(1):39–71.

Collins, M. (2003). Head-driven statistical models for natural language parsing. *Computational Linguistics*, 29:589–637.

Della Pietra, S., Della Pietra, V., and Lafferty, J. (1997). Inducing features of random fields. *IEEE Transactions on Pattern Analysis and Machine Intelligence*, 19(4):380–393.

Genzel, D. and Charniak, E. (2002). Entropy rate constancy in text. In *Proceedings of ACL*.

Hale, J. (2001). A probabilistic Earley parser as a psycholinguistic model. In *Proceedings of NAACL*, volume 2, pages 159–166.

Jaeger, T. F. (2005). Optional *that* indicates production difficulty: evidence from disfluencies. In *Proceedings of Disfluency in Spontaneous Speech Workshop*.

Jaeger, T. F. (2006). *Redundancy and Syntactic Reduction in Spontaneous Speech*. PhD thesis, Stanford University, Stanford, CA.

Keller, F. (2004). The entropy rate principle as a predictor of processing effort: An evaluation against eye-tracking data. In *Proceedings of the Conference on Empirical Methods in Natural Language Processing*, pages 317–324, Barcelona.

Levy, R. (2005). *Probabilistic Models of Word Order and Syntactic Discontinuity*. PhD thesis, Stanford University.

Levy, R. (2006). Expectation-based syntactic comprehension. Ms., University of Edinburgh.

Roark, B. (2001). Probabilistic top-down parsing and language modeling. *Computational Linguistics*, 27(2):249–276.

Roland, D., Elman, J. L., and Ferreira, V. S. (2006). Why is that? Structural prediction and ambiguity resolution in a very large corpus of English sentences. *Cognition*, 98:245–272.

Wasow, T., Jaeger, T. F., and Orr, D. (in press). Lexical variation in relativizer frequency. In Wiese, H. and Simon, H., editors, *Proceedings of the Workshop on Expecting the unexpected: Exceptions in Grammar at the 27th Annual Meeting of the German Linguistic Association*, University of Cologne, Germany. DGfS.
